# Causal discovery with scale-mixture model for spatiotemporal variance dependencies

**Zhitang Chen**[*], **Kun Zhang**[†], and **Laiwan Chan**[*]

[*]Department of Computer Science and Engineering, Chinese University of Hong Kong, Hong Kong
{ztchen,lwchan}@cse.cuhk.edu.hk
[†]Max Planck Institute for Intelligent Systems, Tübingen, Germany
kzhang@tuebingen.mpg.de

## Abstract

In conventional causal discovery, structural equation models (SEM) are directly applied to the observed variables, meaning that the causal effect can be represented as a function of the direct causes themselves. However, in many real world problems, there are significant dependencies in the variances or energies, which indicates that causality may possibly take place at the level of variances or energies. In this paper, we propose a probabilistic causal scale-mixture model with spatiotemporal variance dependencies to represent a specific type of generating mechanism of the observations. In particular, the causal mechanism including contemporaneous and temporal causal relations in variances or energies is represented by a Structural Vector AutoRegressive model (SVAR). We prove the identifiability of this model under the non-Gaussian assumption on the innovation processes. We also propose algorithms to estimate the involved parameters and discover the contemporaneous causal structure. Experiments on synthetic and real world data are conducted to show the applicability of the proposed model and algorithms.

## 1 Introduction

Causal discovery aims to discover the underlying generating mechanism of the observed data, and consequently, the causal relations allow us to predict the effects of interventions on the system [15, 19]. For example, if we know the causal structure of a stock market, we are able to predict the reactions of other stocks against the sudden collapse of one share price in the market. A traditional way to infer the causal structure is by controlled experiments. However, controlled experiments are in general expensive, time consuming, technically infeasible and/or ethically prohibited. Thus, causal discovery from non-experimental data is of great importance and has drawn considerable attention in the past decades [15, 19, 16, 17, 12, 22, 2]. Probabilistic models such as Bayesian Networks (BNs) and Linear Non-Gaussian Acyclic Models (LiNGAM) have been proposed and applied to many real world problems [18, 13, 14, 21].

Conventional models such as LiNGAM assume that the causal relations are of a linear form, i.e., if the observed variable $x$ is the cause of another observed variable $y$, we model the causal relation as $y = \alpha x + e$, where $\alpha$ is a constant coefficient and $e$ is the additive noise independent of $x$. However, in many types of natural signals or time series such as MEG/EEG data [23] and financial data [20], a common form of nonlinear dependency, as seen from the correlation in variances or energies, is found [5]. This observation indicates that causality may take place at the level of variances or energies instead of the observed variables themselves. Generally speaking, traditional methods cannot detect this type of causal relations. Another restriction of conventional causal models is that these models assume constant variances of the observations; this assumption is unrealistic for those data with strong heteroscedasticity [1].

In this paper, we propose a new probabilistic model called Causal Scale-Mixture model with SpatioTemporal Variance Dependencies (CSM-STVD) incorporating the spatial and temporal variance or energy dependencies among the observed data. The main feature of the new model is that we model the spatiotemporal variance dependencies based on the Structural Vector AutoRegressive (SVAR) model, in particular the Non-Gaussian SVAR [11]. The contributions of this study are two-fold. First, we provide an alternative way to model the causal relations among the observations, i.e., causality in variances or energies. In this model, causality takes place at the level of variances or energies, i.e., the variance or energy of one observed series at time instant $t_0$ is influenced by the variances or energies of other variables at time instants $t \leq t_0$ and its past values at time instants $t < t_0$. Thus, both contemporaneous and temporal causal relations in variances are considered. Secondly, we prove the identifiability of this model and more specifically, we show that Non-Gaussianity makes the model fully identifiable. Furthermore, we propose a method which directly estimates such causal structures without explicitly estimating the variances.

## 2   Related work

To model the variance or energy dependencies of the observations, a classic method is to use a scale-mixture model [5, 23, 9, 8]. Mathematically, we can represent a signal as $s_i = u_i \sigma_i$, where $u_i$ is a signal with zero mean and constant variance, and $\sigma_i$ is a positive factor which is independent of $u_i$ and modulates the variance or energy of $s_i$ [5]. For multivariate case, we have

$$\mathbf{s} = \mathbf{u} \odot \boldsymbol{\sigma}, \tag{1}$$

where $\odot$ means element-wise multiplication. In basic scale-mixture model, $\mathbf{u}$ and $\boldsymbol{\sigma}$ are statistically independent and the components $u_i$ are spatiatemporally independent, i.e. $u_{i,t_{\tau_1}} \perp\!\!\!\perp u_{j,t_{\tau_2}}, \forall t_{\tau_1}, t_{\tau_2}$. However, $\sigma_i$, the standard deviations of the observations, are dependent across $i$. The observation $\mathbf{x}$, in many situations, is assumed to be a linear mixture of the source $\mathbf{s}$, i.e., $\mathbf{x} = \mathbf{As}$, where $\mathbf{A}$ is a mixing matrix.

In [5], Hirayama and Hyvärinen proposed a two-stage model. The first stage is a classic ICA model [3, 10], where the observation $\mathbf{x}$ is a linear mixture of the hidden source $\mathbf{s}$, i.e., $\mathbf{x} = \mathbf{As}$. On the second stage, the variance dependencies are modeled by applying a linear Non-Gaussian (LiN) SEM to the log-energies of the sources.

$$y_i = \sum_j h_{ij} y_j + h_{i0} + r_i, i = 1, 2, \cdots, d,$$

where $y_i = log\, \phi(\sigma_i)$ are the log-energies of sources $s_i$ and the nonlinear function $\phi$ is any appropriate measure of energy; $r_i$ are non-Gaussian distributed and independent of $y_j$. To make the problem tractable, they assumed that $u_i$ are binary, i.e., $u_i \in \{-1, 1\}$ and uniformly distributed. The parameters of this two-stage model including $\mathbf{A}$ and $h_{ij}$ are estimated by maximum likelihood without approximation due to the uniform binary distribution assumption of $\mathbf{u}$. However, this assumption is restrictive and thus may not fit real world observations well. Furthermore, they assumed that $\sigma_i$ are spatially dependent but temporally white. However, many time series show strong heterosecadasticity and temporal variance dependencies such as financial time series and brain signals. Taking into account of temporal variance dependencies would improve the quality of the estimated underlying structure of the observed data.

Another two-stage model for magnetoencephalography (MEG) or electroencephalography (EEG) data was propsoed earlier in [23]. The first stage also performs linear separation; they proposed a blind source separation algorithm by exploiting the autocorrelations and time-varying variances of the sources. In the second stage, $s_i(t)$ are modeled by autoregressive processes with $L$ lags (AR($L$)) driven by innovations $e_i(t)$. The innovation processes $e_i(t)$ are mutually uncorrelated and temporally white. However, $e_i(t)$ are not necessarily independent. They modeled $e_i(t)$ as follows:

$$e_i(t) = \sigma_{it} z_i(t), \text{ where } z_i(t) \sim \mathcal{N}(0, 1). \tag{2}$$

Two different methods are used to model the dependencies among the variances of the innovations. The first method is causal-in-variance GARCH (CausalVar-GARCH). Specifically $\sigma_{it}^2$ are modeled by a multivariate GARCH model. The advantage of this model is that we are able to estimate the temporal causal structure in variances. However, this model provides no information about the

contemporaneous causal relations among the sources if there indeed exist such causal relations. The second method to model the variance dependencies is applying a factor model to the log-energies ($log\,\sigma_{it}^2$) of the sources. The disadvantage of this method is that we cannot model the causal relations among the sources which are more interesting to us.

In many real world observations, there are causal influences in variances among the observed variables. For instance, there are significant mutual influences among the volatilities of the observed stock prices. We are more interested in investigating the underlying causal structure among the variances of the observed data. Consequently, in this paper, we consider the situation where the correlation in the variances of the observed data is interesting. That is, the first stage of [5, 23] is not needed, and we focus on the second stage, i.e., modeling the spatiotemporal variance dependencies and causal mechanism among the observations. In the following sections, we propose our probabilistic model based on SVAR to describe the spatiotemporal variance dependencies among the observations. Our model is, as shown in later sections, closely related to the models introduced in [5, 23], but has significant advantages: (1) both contemporaneous and temporal causal relations can be modeled; (2) this model is fully identifiable under certain assumptions.

## 3  Causal scale-mixture model with spatiotemporal variances dependencies

We propose the causal scale-mixture model with spatiotemporal variance dependencies as follows. Let $\mathbf{z}(t)$ be the $m \times 1$ observed vector with components $z_i(t)$, which are assumed to be generated according to the scale-mixture model:

$$z_i(t) = u_i(t)\sigma_i(t). \tag{3}$$

Here we assume that $u_i(t)$ are temporally independent processes, i.e., $u_i(t_{\tau_1}) \perp\!\!\!\perp u_j(t_{\tau_2}), \forall t_{\tau_1} \neq t_{\tau_2}$ but unlike basic scale-mixture model, here $u_i(t)$ may be contemporarily dependent, i.e., $u_i(t) \not\!\perp\!\!\!\perp u_j(t), \forall i \neq j$. $\boldsymbol{\sigma}(t)$ is spatially and temporally independent of $\mathbf{u}(t)$. Using vector notation,

$$\mathbf{z}_t = \mathbf{u}_t \odot \boldsymbol{\sigma}_t. \tag{4}$$

Here $\sigma_{it} > 0$ are related to the variances or energies of the observations $\mathbf{z}_t$ and are assumed to be spatiotemporally dependent. As in [5, 23], let $\mathbf{y}_t = log\,\boldsymbol{\sigma}_t$. In this paper, we model the spatiotemporal variance dependencies by a Structural Vector AutoRegressive model (SVAR), i.e.,

$$\mathbf{y}_t = \mathbf{A}_0\mathbf{y}_t + \sum_{\tau=1}^{L} \mathbf{B}_\tau \mathbf{y}_{t-\tau} + \boldsymbol{\epsilon}_t, \tag{5}$$

where $\mathbf{A}_0$ contains the contemporaneous causal strengths among the variances of the observations, i.e., if $[\mathbf{A}_0]_{ij} \neq 0$, we say that $y_{it}$ is contemporaneously affected by $y_{jt}$; $\mathbf{B}_\tau$ contains the temporal (time-lag) causal relations, i.e., if $[\mathbf{B}_\tau]_{ij} \neq 0$, we say that $y_{i,t}$ is affected by $y_{j,t-\tau}$. Here, $\boldsymbol{\epsilon}_t$ are i.i.d. mutually independent innovations. Let $\mathbf{x}_t = log\,|\mathbf{z}_t|$ (In this model, we assume that $u_i(t)$ do not take value zero) and $\boldsymbol{\eta}_t = log\,|\mathbf{u}_t|$. Take $log$ of the absolute values of both sides of equation (4), then we have the following model:

$$\begin{aligned}
\mathbf{x}_t &= \mathbf{y}_t + \boldsymbol{\eta}_t, \\
\mathbf{y}_t &= \mathbf{A}_0\mathbf{y}_t + \sum_{\tau=1}^{L} \mathbf{B}_\tau \mathbf{y}_{t-\tau} + \boldsymbol{\epsilon}_t.
\end{aligned} \tag{6}$$

We make the following assumptions on the model:

$\mathcal{A}_1$ Both $\boldsymbol{\eta}_t$ and $\boldsymbol{\epsilon}_t$ are temporally white with zero means. The components of $\boldsymbol{\eta}_t$ are not necessarily independent, and we assume that the covariance matrix of $\boldsymbol{\eta}_t$ is $\boldsymbol{\Sigma}_{\boldsymbol{\eta}}$. The components of $\boldsymbol{\epsilon}_t$ are independent and $\boldsymbol{\Sigma}_{\boldsymbol{\epsilon}} = \mathbf{I}$[1].

$\mathcal{A}_2$ The contemporaneous causal structure is acyclic, i.e., by simultaneous row and column permutations, $\mathbf{A}_0$ can be permuted to a strictly lower triangular matrix. $\mathbf{B}_L$ is of full rank.

$\mathcal{A}_3$ The innovations $\boldsymbol{\epsilon}_t$ are non-Gaussian, and $\boldsymbol{\eta}_t$ are either Gaussian or non-Gaussian.

Inspired by the identifiability results of the Non-Gaussian state-space model in [24], we show that our model is identifiable. Note that our new model and the state-space model proposed in [24] are two different models, while interestingly by simple re-parameterization we can prove the following Lemma 3.1 and Theorem 3.1 following [24].

**Lemma 3.1** *Given the log-transformed observation $\mathbf{x}_t = log\,|\mathbf{z}_t|$ generated by Equations (6), if the assumptions $\mathcal{A}_1 \sim \mathcal{A}_2$ hold, by solving simple linear equations involving the autocovariances of $\mathbf{x}_t$, the covariance $\boldsymbol{\Sigma}_{\boldsymbol{\eta}}$ and $\mathbf{AB}_\tau$ can be uniquely determined, where $\mathbf{A} = (\mathbf{I} - \mathbf{A}_0)^{-1}$; furthermore, $\mathbf{A}$ and $\mathbf{B}_\tau$ can be identified up to some rotation transformations. That is, suppose that two models with parameters $(\mathbf{A}, \{\mathbf{B}_\tau\}_{\tau=1}^L, \boldsymbol{\Sigma}_{\boldsymbol{\eta}})$ and $(\tilde{\mathbf{A}}, \{\tilde{\mathbf{B}}_\tau\}_{\tau=1}^L, \tilde{\boldsymbol{\Sigma}}_{\tilde{\boldsymbol{\eta}}})$ generate the same observation $\mathbf{x}_t$, then we have $\boldsymbol{\Sigma}_{\boldsymbol{\eta}} = \tilde{\boldsymbol{\Sigma}}_{\tilde{\boldsymbol{\eta}}}$, $\tilde{\mathbf{A}} = \mathbf{AU}$, $\tilde{\mathbf{B}}_\tau = \mathbf{U}^T \mathbf{B}_\tau$, where $\mathbf{U}$ is an orthogonal matrix.*

Non-Gaussianity of the innovations $\boldsymbol{\epsilon}_t$ makes the model fully identifiable, as seen in the following theorem.

**Theorem 3.1** *Given the log-transformed observation $\mathbf{x}_t = log\,|\mathbf{z}_t|$ generated by Equations (6) and given L, if assumptions $\mathcal{A}_1 \sim \mathcal{A}_3$ hold, then the model is identifiable. In other words, suppose that two models with parameters $(\mathbf{A}, \{\mathbf{B}_\tau\}_{\tau=1}^L, \boldsymbol{\Sigma}_{\boldsymbol{\eta}})$ and $(\tilde{\mathbf{A}}, \{\tilde{\mathbf{B}}_\tau\}_{\tau=1}^L, \tilde{\boldsymbol{\Sigma}}_{\tilde{\boldsymbol{\eta}}})$ generate the same observation $\mathbf{x}_t$; then these two models are identical, i.e., we have $\tilde{\boldsymbol{\Sigma}}_{\tilde{\boldsymbol{\eta}}} = \boldsymbol{\Sigma}_{\boldsymbol{\eta}}$, $\tilde{\mathbf{A}} = \mathbf{A}$, $\tilde{\mathbf{B}}_\tau = \mathbf{B}_\tau$, and $\tilde{\mathbf{y}}_t = \mathbf{y}_t$.*

## 4 Parameter learning and causal discovery

In this section, we propose an effective algorithm to estimate the contemporaneous causal structure matrix $\mathbf{A}_0$ and temporal causal structure matrices $\mathbf{B}_\tau$, $\tau = 1, \cdots, L$ (see (6)).

### 4.1 Estimation of $\mathbf{AB}_\tau$

We have shown that $\mathbf{AB}_\tau$ can be uniquely determined, where $\mathbf{A} = (\mathbf{I} - \mathbf{A}_0)^{-1}$. The proof of Lemma 3.1 also suggests a way to estimate $\mathbf{AB}_\tau$, as given below. Readers can refer to the appendix for the detailed mathematical derivation. Although we are aware that this method might not be statistically efficient, we adopt this estimation method due to its great computational efficiency. Given the log-transformed observations $\mathbf{x}_t = log\,|\mathbf{z}_t|$, denoted by $\mathbf{R}_{\mathbf{x}}(k)$ the autocovariance function of $\mathbf{x}_t$ at lag $k$, we have $\mathbf{R}_{\mathbf{x}}(k) = \mathbb{E}(\mathbf{x}_t \mathbf{x}_{t+k}^T)$. Based on the model assumptions $\mathcal{A}_1$ and $\mathcal{A}_2$, we have the following linear equations of the autocovarainces of $\mathbf{x}_t$.

$$
\begin{bmatrix} \mathbf{R}_{\mathbf{x}}(L+1) \\ \mathbf{R}_{\mathbf{x}}(L+2) \\ \vdots \\ \mathbf{R}_{\mathbf{x}}(2L) \end{bmatrix} = \underbrace{\begin{bmatrix} \mathbf{R}_{\mathbf{x}}(L) & \mathbf{R}_{\mathbf{x}}(L-1) & \cdots & \mathbf{R}_{\mathbf{x}}(1) \\ \mathbf{R}_{\mathbf{x}}(L+1) & \mathbf{R}_{\mathbf{x}}(L) & \cdots & \mathbf{R}_{\mathbf{x}}(2) \\ \vdots & \vdots & \ddots & \vdots \\ \mathbf{R}_{\mathbf{x}}(2L-1) & \mathbf{R}_{\mathbf{x}}(2L-2) & \cdots & \mathbf{R}_{\mathbf{x}}(L) \end{bmatrix}}_{\triangleq \mathbf{H}} \begin{bmatrix} \mathbf{C}_1^T \\ \mathbf{C}_2^T \\ \vdots \\ \mathbf{C}_L^T \end{bmatrix}, \tag{7}
$$

where $\mathbf{C}_\tau = \mathbf{AB}_\tau (\tau = 1, \cdots, L)$. As shown in the proof of Lemma 3.1, $\mathbf{H}$ is invertible. We can easily estimate $\mathbf{AB}_\tau$ by solving the linear Equations (7).

### 4.2 Estimation of $\mathbf{A}_0$

The estimations of $\mathbf{AB}_\tau (\tau = 1, \cdots, L)$ still contain the mixing information of the causal structures $\mathbf{A}_0$ and $\mathbf{B}_\tau$. In order to further obtain the contemporaneous and temporal causal relations, we need to estimate both $\mathbf{A}_0$ and $\mathbf{B}_\tau (\tau = 1, \cdots, L)$. Here, we show that the estimation of $\mathbf{A}_0$ can be reduced to solving a Linear Non-Gaussian Acyclic Models with latent confounders.

Substituting $\mathbf{y}_t = \mathbf{x}_t - \boldsymbol{\eta}_t$ into Equations (6), we have

$$
\mathbf{x}_t - \boldsymbol{\eta}_t = \sum_{\tau=1}^L \mathbf{AB}_\tau (\mathbf{x}_{t-\tau} - \boldsymbol{\eta}_{t-\tau}) + \mathbf{A}\boldsymbol{\epsilon}_t. \tag{8}
$$

Since $\mathbf{AB}_\tau$ can be uniquely determined according to Lemma 3.1 or more specifically Equations (7), we can easily obtain $\boldsymbol{\xi}_t = \mathbf{x}_t - \sum_{\tau=1}^{L} \mathbf{AB}_\tau \mathbf{x}_{t-\tau}$, then we have:

$$\boldsymbol{\xi}_t = \mathbf{A}\boldsymbol{\epsilon}_t + \boldsymbol{\eta}_t - \sum_{\tau=1}^{L} \mathbf{AB}_\tau \boldsymbol{\eta}_{t-\tau}. \tag{9}$$

This is exactly a Linear Non-Gaussian Acyclic Model with latent confounders and the estimation of $\mathbf{A}$ is a very challenging problem [6, 2]. To make to problem tractable, we further have the following two assumptions on the model:

- $\mathcal{A}_4$ If the components of $\boldsymbol{\eta}_t$ are not independent, we assume that $\boldsymbol{\eta}_t$ follows a factor model: $\boldsymbol{\eta}_t = \mathbf{Df}_t$, where the components of $\mathbf{f}_t$ are spatially and temporally independent Gaussian factors and $\mathbf{D}$ is the factor loading matrix (not necessarily square).

- $\mathcal{A}_5$ The components of $\boldsymbol{\epsilon}_t$ are simultaneously super-Gaussian or sub-Gaussian.

By replacing $\boldsymbol{\eta}_t$ with $\mathbf{Df}_t$ , we have:

$$\boldsymbol{\xi}_t = \mathbf{A}\boldsymbol{\epsilon}_t + \underbrace{\mathbf{Df}_t - \sum_{\tau=1}^{L} \mathbf{AB}_\tau \mathbf{Df}_{t-\tau}}_{\text{confounding effects}} . \tag{10}$$

To identify the matrix $\mathbf{A}$ which contains the contemporaneous causal information of the observed variables, we treat $\boldsymbol{f}_t$ and $\boldsymbol{f}_{t-\tau}$ as latent confounders and the interpretation of assumption $\mathcal{A}_4$ is that we can treat the independent factors $\mathbf{f}_t$ as some external factors outside the system. The Gaussian assumption of $\mathbf{f}_t$ can be interpreted hierarchically as the result of central limit theorem because these factors themselves represent the ensemble effects of numerous factors from the whole environment. On the contrary, the disturbances $\epsilon_{it}$ are local factors that describe the intrinsic behaviors of the observed variables [4]. Since they are local and thus not regarded as the ensembles of large amount of factors. In this case, the disturbances $\epsilon_{it}$ are assumed to be non-Gaussian.

The LiNGAM-GC model [2] takes into the consideration of latent confounders. In that model, the confounders are assumed to follow Gaussian distribution, which was interpreted as the result of central limit theorem. It mainly focuses on the following cause-effect pair:

$$\begin{aligned} x &= e_1 + \alpha c, \\ y &= \rho x + e_2 + \beta c, \end{aligned} \tag{11}$$

where $e_1$ and $e_2$ are non-Gaussian and mutually independent, and $c$ is the latent Gaussian confounder independent of the disturbances $e_1$ and $e_2$. To tackle the causal discovery problem of LiNGAM-GC, it was firstly shown that if $x$ and $y$ are standardized to unit absolute kurtosis then $|\rho| < 1$ based on the assumption that $e_1$ and $e_2$ are simultaneously super-Gaussian or sub-Gaussian. Note that assumption $\mathcal{A}_5$ is a natural extension of this assumption. It holds in many practical problems, especially for financial data. After the standardization, the following cumulant-based measure $\tilde{R}_{xy}$ was proposed [2]:

$$\begin{aligned} \tilde{R}_{xy} &= (C_{xy} + C_{yx})(C_{xy} - C_{yx}), \quad \text{where} \\ C_{xy} &= \hat{\mathbb{E}}\{x^3 y\} - 3\hat{\mathbb{E}}\{xy\}\hat{\mathbb{E}}\{x^2\}, \\ C_{yx} &= \hat{\mathbb{E}}\{xy^3\} - 3\hat{\mathbb{E}}\{xy\}\hat{\mathbb{E}}\{y^2\}, \end{aligned} \tag{12}$$

and $\hat{\mathbb{E}}$ means sample average. It was shown that the causal direction can be identified simply by examining the sign of $\tilde{R}_{xy}$, i.e., if $\tilde{R}_{xy} > 0$, $x \to y$ is concluded; otherwise if $\tilde{R}_{xy} < 0$, $y \to x$ is concluded. Once the causal direction has been identified, the estimation of causal strength is straightforward. The work can be extended to multivariate causal network discovery following DirectLiNGAM framework [17].

Here we adopt LiNGAM-GC-UK, the algorithm proposed in [2], to find the contemporaneous casual structure matrix $\mathbf{A}_0$. Once $\mathbf{A}_0$ has been estimated, $\mathbf{B}_\tau$ can be easily obtained by $\hat{\mathbf{B}}_\tau = (\mathbf{I} - \hat{\mathbf{A}}_0)\hat{\mathbf{C}}_\tau$, where $\hat{\mathbf{A}}_0$ and $\hat{\mathbf{C}}_\tau$ are the estimations of $\mathbf{A}_0$ and $\mathbf{AB}_\tau$, respectively. The algorithm for learning the model is summarized in the following algorithm.

**Algorithm 1** Causal discovery with scale-mixture model for spatiotemporal variance dependencies

1: Given the observations $\mathbf{z}_t$, compute $\mathbf{x}_t = log\,|\mathbf{z}_t|$.
2: Subtract the mean $\bar{\mathbf{x}}_t$ from $\mathbf{x}_t$, i.e., $\mathbf{x}_t = \mathbf{x}_t - \bar{\mathbf{x}}_t$
3: Choose an appropriate lag $L$ for the SVAR and then estimate $\mathbf{AB}_\tau$ where $\mathbf{A} = (\mathbf{I} - \mathbf{A}_0)^{-1}$ and $\tau = 1, \cdots, L$, using Equations(7).
4: Obtain the residues by $\boldsymbol{\xi}_t = \mathbf{x}_t - \sum_{\tau=1}^{L} \mathbf{AB}_\tau \mathbf{x}_{t-\tau}$.
5: Apply LiNGAM-GC algorithms to $\boldsymbol{\xi}_t$ and obtain the estimation of $\mathbf{A}_0$ and $\mathbf{B}_\tau (\tau = 1, \cdots, L)$ and the corresponding comtemporaneous and temporal causal orderings.

## 5 Experiment

We conduct experiments using synthetic data and real world data to investigate the effectiveness of our proposed model and algorithms.

### 5.1 Synthetic data

We generate the observations according to the following model:

$$\mathbf{z}_t = \mathbf{r} \odot \mathbf{u}_t \odot \boldsymbol{\sigma}_t,$$

$\mathbf{r}$ is a $m \times 1$ scale vector of which the elements are randomly selected from interval $[1.0, 6.0]$; $\mathbf{u}_t > 0$ and $\boldsymbol{\eta}_t = log\,\mathbf{u}_t$ follows a factor model:

$$\boldsymbol{\eta}_t = \mathbf{D}\mathbf{f}_t,$$

where $\mathbf{D}$ is $m \times m$ and the elements of $\mathbf{D}$ are randomly selected from $[0.2, 0.4]$ . $f_{it}$ are i.i.d. and $f_{it} \sim \mathcal{N}(0, 0.5)$. Denoted by $\mathbf{y}_t = log\,\boldsymbol{\sigma}_t$, we model the spatiotemporal variance dependencies of the observations $\mathbf{x}_t$ by an SVAR(1):

$$\mathbf{y}_t = \mathbf{A}_0 \mathbf{y}_t + \mathbf{B}_1 \mathbf{y}_{t-1} + \boldsymbol{\epsilon}_t,$$

where $\mathbf{A}_0$ is a $m \times m$ strictly lower triangular matrix of which the elements are randomly selected from $[0.1, 0.2]$ or $[-0.2, -0.1]$; $\mathbf{B}_1$ is a $m \times m$ matrix of which the diagonal elements $[\mathbf{B}_1]_{ii}$ are randomly selected from $[0.7, 0.8]$, 80% of the off-diagonal elements $[\mathbf{B}_1]_{i \neq j}$ are zero and the remaining 20% are randomly selected from $[-0.1, 0.1]$; $\epsilon_{it}$ are i.i.d. super-Gaussian generated by $\epsilon_{it} = sign(n_{it})|n_{it}|^2 (n_{it} \sim \mathcal{N}(0,1))$ and normalized to unit variance. The generated observations are permuted to a random order. The task of this experiment is to investigate the performance of our algorithms in estimating the coefficient matrix $(\mathbf{I} - \mathbf{A}_0)^{-1}\mathbf{B}_1$ and also the contemporaneous causal ordering induced by $\mathbf{A}_0$. We estimate the matrix $(\mathbf{I} - \mathbf{A}_0)^{-1}\mathbf{B}_1$ using Lemma 3.1 or specifically Equations (7). We use different algorithms: LiNGAM-GC-UK proposed in [2], C-M proposed in [7] and LiNGAM [16] to estimate the contemporaneous causal structure. We investigate the performances of different algorithms in the scenarios of $m = 4$ with sample size from 500 to 4000 and $m = 8$ with sample size from 1000 to 10000. For each scenario, we randomly conduct 100 independent trials and discard those trials where the SVAR processes are not stable. We calculate the accuracies of LiNGAM-GC-UK, C-M and LiNGAM in finding (1) whole causal ordering (2) exogenous variable (root) of the causal network. We also calculate the sum square error $Err$ of estimated causal strength matrix of different algorithms with respect to the true one. The average SNR defined as $SNR = 10\,log\,\frac{\sum_i Var(\epsilon_i)}{\sum_i Var(f_i)}$ is about 13.85 dB. The experimental results are shown in Figure 1 and Table 1. Figure 1 shows the plots of the estimated entries of $(\mathbf{I} - \mathbf{A}_0)^{-1}\mathbf{B}_1$ versus the true ones when the dimension of the observations $m = 8$. From Figure 1, we can see that the matrix $(\mathbf{I} - \mathbf{A}_0)^{-1}\mathbf{B}_1$ is estimated well enough when the sample size is only 1000. This confirms the correctness of our theoretical analysis of the proposed model. From Table 1, we can see that when the dimension of the observations is small ($m = 4$), all algorithms have acceptable performances. The performance of LiNGAM is the best when the sample size is small. This is because C-M and LiNGAM-GC-UK are cumulant-based methods which need sufficiently large sample size. When the dimension of the observations $m$ increases to 8, we can see that the performances of C-M and LiNGAM degrade dramatically. While LiNGAM-GC-UK still successfully finds the exogenous variable (root) or even the whole contemporaneous causal ordering among the variances of the observations if the sample size is sufficiently large enough. This is mainly due to the fact that when the dimension increases,

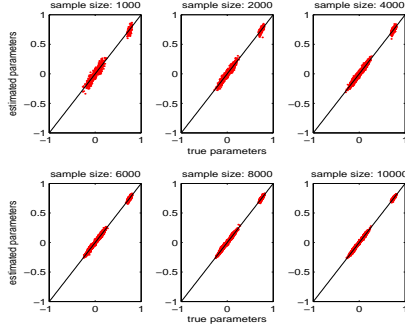

Figure 1: Estimated entries causal strength matrix $(\mathbf{I} - \mathbf{A}_0)^{-1}\mathbf{B}_1$ vs the true ones $(m = 8)$

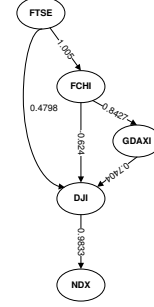

Figure 2: Contemporaneous causal network of the selected stock indices

Table 1: Accuracy of finding the causal ordering

| sample size | whole causal ordering | | | first variable found | | | Err | | |
|---|---|---|---|---|---|---|---|---|---|
| | C-M | LiNGAM | LiNGAM-GC-UK | C-M | LiNGAM | LiNGAM-GC-UK | C-M | LiNGAM | LiNGAM-GC-UK |
| $m = 4$ | | | | | | | | | |
| 500 | 37% | **70%** | 28% | 61% | **85%** | 60% | 0.1101 | **0.0326** | 0.0938 |
| 1000 | 47% | **75%** | 25% | 25% | **92%** | 72% | 0.0865 | **0.024** | 0.0444 |
| 2000 | 74% | **86%** | 81% | 82% | 90% | **92%** | 0.0679 | 0.02 | **0.0199** |
| 3000 | 67% | 78% | **90%** | 79% | 88% | **96%** | 0.0716 | 0.0201 | **0.0126** |
| 4000 | 63% | 83% | **90%** | 81% | 92% | **94%** | 0.0669 | 0.0193 | **0.0109** |
| $m = 8$ | | | | | | | | | |
| 1000 | 0% | **23.08%** | 8.79% | 20.88% | **75.82%** | 65.93% | 0.8516 | **0.2318** | 0.3017 |
| 2000 | 1.14% | **26.14%** | 25% | 25% | 70.45% | 75% | 0.7866 | 0.2082 | **0.1396** |
| 4000 | 0% | 31.87% | **58.24%** | 19.78% | 82.41% | **86.81%** | 0.7537 | 0.1916 | **0.0634** |
| 6000 | 0% | 25.29% | **83.91%** | 25.29% | 75.86% | **96.55%** | 0.7638 | 0.1843 | **0.0341** |
| 8000 | 2.20% | 30.77% | **80.22%** | 17.58% | 79.12% | **91.21%** | 0.7735 | 0.1824 | **0.029** |
| 10000 | 0% | 23.53% | **91.76%** | 12.94% | 68.24% | **97.64%** | 0.7794 | 0.194 | **0.0199** |

the confounding effects of $\mathbf{Df}_t - (\mathbf{I} - \mathbf{A})^{-1}\mathbf{B}_1\mathbf{Df}_{t-1}$ become more problematic such that the performances of C-M and LiNGAM are strongly affected by confounding effect. Table 1 also shows the estimation accuracies of the compared methods. Among them, LiNGAM-GC-UK significantly outperforms other methods given sufficiently large sample size.

In order to investigate the robustness of our methods against the Gaussian assumption on the external factors $\mathbf{f}_t$, we conduct the following experiment. The experimental setting is the same as that in the above experiment but here the external factors $\mathbf{f}_t$ are non-Gaussian, and more specifically $f_{it} = sign(n_{it})|n_{it}|^p$, where $n_{it} \sim \mathcal{N}(0, 0.5)$. When $p > 1$, the factor is super-Gaussian and when $p < 1$ the factor is sub-Gaussian. We investigate the performances of LiNGAM-GC-UK, LiNGAM and C-M in finding the whole causal ordering in difference scenarios where $p = \{0.4, 0.6, 0.8, 1.0, 1.2, 1.4, 1.6\}$ with sample size of 6000. The results in Figure 3 show that LiNGAM-GC-UK achieved satisfying results compared to LiNGAM and C-M. This suggests that although LiNGAM-GC is developed based on the assumption that the latent confounders are Gaussian distributed, it is still robust in the scenarios where the latent confounders are mildly non-Gaussian with mild causal strength

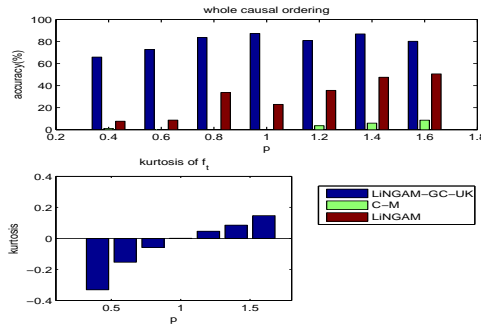

Figure 3: Robustness against Gaussianity of $\mathbf{f}_t$

## 5.2 Real world data

In this section, we use our new model to discover the causal relations among five major world stocks indices: (1) Dow Jones Industrial Average (DJI) (2) FTSE 100 (FTSE) (3) Nasdaq-100 (NDX) (4) CAC 40 (FCHI) (5) DAX (GDAXI), where DJI and NDX are stock indices in US, and FTSE, FCHI and GDAXI are indices in Europe. Note that because of the time difference, we believe that the causal relations among these stock indices are mainly acyclic, as we assumed in this paper. We collect the adjusted close prices of these selected indices from May 2nd, 2006 to April 12th, 2012, and use linear interpolation to estimate the prices on those dates when the data are not available. We apply our proposed model with SVAR(1) to model the spatiotemporal variance dependencies of the data. For the contemporaneous causal structure discovery, we use LiNGAM-GC-UK, C-M, LiNGAM[2] and Direct-LiNGAM[3] to estimate the causal ordering. The discovered causal orderings of different algorithms are shown in Table 2. From Table 2, we see that in the causal ordering

Table 2: Contemporaneous causal ordering of the selected stock indices

| algorithm | causal ordering |
|---|---|
| LiNGAM-GC-UK | $\{2\} \rightarrow \{4\} \rightarrow \{5\} \rightarrow \{1\} \rightarrow \{3\}$ |
| C-M | $\{1\} \rightarrow \{2\} \rightarrow \{4\} \rightarrow \{5\}$ |
| | $\{1\} \rightarrow \{3\}$ |
| LiNGAM | $\{2\} \rightarrow \{5\} \rightarrow \{3\} \rightarrow \{1\}$ |
| | $\{2\} \rightarrow \{4\}$ |
| Direct-LiNGAM | $\{3\} \rightarrow \{1\} \rightarrow \{5\} \rightarrow \{4\} \rightarrow \{2\}$ |

discovered by LiNGAM-GC-UK and LiNGAM, the stock indices in US, i.e., DJI and NDX are contemporaneously affected by the indices in Europe. Note that each stock index is given in local time. Because of the time difference between Europe and America and the efficient market hypothesis (the market is quick to absorb new information and adjust stock prices relative to that), the contemporaneous causal relations should be from Europe to America, if they exist. This is consistent with the results our method and LiNGAM produced. Another interesting finding is that in the graphs obtained by LiNGAM-GC-UK and LiNGAM, we can see that FTSE is the root, which is consistent with the fact that London is the financial centre of Europe and FTSE is regarded as Europe's most important index. However, in results by C-M and DirectLiNGAM, we have the opposite direction, i.e., the stock indices in US is contemporaneously the cause of the indices in Europe, which is difficult to interpret. The contemporaneous causal network of the stock indices are shown in Figure 2. Further interpretation on the discovered causal strengths needs expertise knowledge.

## 6 Conclusion

In this paper, we investigate the causal discovery problem where causality takes place at the level of variances or energies instead of the observed variables themselves. We propose a causal scale-mixture model with spatiotemporal variance dependencies to describe this type of causal mechanism. We show that the model is fully identifiable under the non-Gaussian assumption of the innovations. In addition, we propose algorithms to estimate the parameters, especially the contemporaneous causal structure of this model. Experimental results on synthetic data verify the practical usefulness of our model and the effectiveness of our algorithms. Results using real world data further suggest that our new model can possibly explain the underlying interaction mechanism of major world stock markets.

**Acknowledgments**

The work described in this paper was partially supported by a grant from the Research Grants Council of the Hong Kong Special Administration Region, China.

## Footnotes

[1]Note that $\boldsymbol{\Sigma}_{\boldsymbol{\epsilon}} = \mathbf{I}$ is assumed just for convenience. $\mathbf{A}_0$ and $\mathbf{B}_\tau$ can also be correctly estimated if $\boldsymbol{\Sigma}_{\boldsymbol{\epsilon}}$ is a general diagonal covariance matrix. The explanation why the scaling indeterminacy can be eliminated is the same as LiNGAM given in [16].

[2]LiNGAM converges to several local optima. We only show one of the discovered causal ordering here. The code is available at:http://www.cs.helsinki.fi/group/neuroinf/lingam/

[3]http://www.ar.sanken.osaka-u.ac.jp/~inazumi/dlingam.html

# References

[1] T. Bollerslev. Generalized autoregressive conditional heteroskedasticity. *Journal of econometrics*, 31(3):307–327, 1986.

[2] Z. Chen and L. Chan. Causal discovery for linear non-gaussian acyclic models in the presence of latent gaussian confounders. In *Proceedings of the 10th international conference on Latent Variable Analysis and Signal Separation*, pages 17–24. Springer-Verlag, 2012.

[3] P. Comon. Independent component analysis, a new concept? *Signal processing*, 36(3):287–314, 1994.

[4] R. Henao and O. Winther. Sparse linear identifiable multivariate modeling. *Journal of Machine Learning Research*, 12:863–905, 2011.

[5] J. Hirayama and A. Hyvärinen. Structural equations and divisive normalization for energy-dependent component analysis. *Advances in Neural Information Processing Systems (NIPS2011)*, 24, 2012.

[6] P.O. Hoyer, S. Shimizu, A.J. Kerminen, and M. Palviainen. Estimation of causal effects using linear non-gaussian causal models with hidden variables. *International Journal of Approximate Reasoning*, 49(2):362–378, 2008.

[7] A. Hyvärinen. Pairwise measures of causal direction in linear non-gaussian acyclic models. In *JMLR Workshop and Conference Proceedings (Proc. 2nd Asian Conference on Machine Learning), ACML2010*, volume 13, pages 1–16, 2010.

[8] A. Hyvärinen, P. O. Hoyer, and M. Inki. Topographic independent component analysis. *Neural Computation*, 13(7):1527–1558, 2001.

[9] A. Hyvärinen and J. Hurri. Blind separation of sources that have spatiotemporal variance dependencies. *Signal Processing*, 84(2):247–254, 2004.

[10] A. Hyvärinen and E. Oja. Independent component analysis: algorithms and applications. *Neural networks*, 13(4-5):411–430, 2000.

[11] A. Hyvärinen, K. Zhang, S. Shimizu, and P. O. Hoyer. Estimation of a structural vector autoregression model using non-gaussianity. *Journal of Machine Learning Research*, 11:1709–1731, 2010.

[12] D. Janzing, J. Mooij, K. Zhang, J. Lemeire, J. Zscheischler, P. Daniušis, B. Steudel, and B. Schölkopf. Information-geometric approach to inferring causal directions. *Artificial Intelligence*, 2012.

[13] Y. Kawahara, S. Shimizu, and T. Washio. Analyzing relationships among arma processes based on non-gaussianity of external influences. *Neurocomputing*, 2011.

[14] A. Moneta, D. Entner, PO Hoyer, and A. Coad. Causal inference by independent component analysis with applications to micro-and macroeconomic data. *Jena Economic Research Papers*, 2010:031, 2010.

[15] J. Pearl. *Causality: models, reasoning, and inference*. Cambridge Univ Pr, 2000.

[16] S. Shimizu, P.O. Hoyer, A. Hyvärinen, and A. Kerminen. A linear non-gaussian acyclic model for causal discovery. *Journal of Machine Learning Research*, 7:2003–2030, 2006.

[17] S. Shimizu, T. Inazumi, Y. Sogawa, A. Hyvärinen, Y. Kawahara, T. Washio, P.O. Hoyer, and K. Bollen. Directlingam: A direct method for learning a linear non-gaussian structural equation model. *Journal of Machine Learning Research*, 12:1225–1248, 2011.

[18] Y. Sogawa, S. Shimizu, T. Shimamura, A. Hyvärinen, T. Washio, and S. Imoto. Estimating exogenous variables in data with more variables than observations. *Neural Networks*, 2011.

[19] P. Spirtes, C.N. Glymour, and R. Scheines. *Causation, prediction, and search*. The MIT Press, 2000.

[20] K. Zhang and L. Chan. Efficient factor garch models and factor-dcc models. *Quantitative Finance*, 9(1):71–91, 2009.

[21] K. Zhang and L.W. Chan. Extensions of ica for causality discovery in the hong kong stock market. In *Proc. of the 13th international conference on Neural information processing-Volume Part III*, pages 400–409. Springer-Verlag, 2006.

[22] K. Zhang and A. Hyvärinen. On the identifiability of the post-nonlinear causal model. In *Proceedings of the Twenty-Fifth Conference on Uncertainty in Artificial Intelligence*, pages 647–655, 2009.

[23] K. Zhang and A. Hyvärinen. Source separation and higher-order causal analysis of meg and eeg. In *Proceedings of the Twenty-Sixth Conference on Uncertainty in Artificial Intelligence*, pages 709–716, 2010.

[24] K. Zhang and A. Hyvärinen. A general linear non-gaussian state-space model: Identifiability, identification, and applications. In *Proceedings of Asian Conference on Machine Learning, JMLR W&CP*, pages 113–128, 2011.

